# Factorization with uncertainty and missing data: exploiting temporal coherence

**Amit Gruber and Yair Weiss**
School of Computer Science and Engineering
The Hebrew University of Jerusalem
91904 Jerusalem, Israel
{*amitg,yweiss*}*@cs.huji.ac.il*

## Abstract

The problem of "Structure From Motion" is a central problem in vision: given the $2D$ locations of certain points we wish to recover the camera motion and the $3D$ coordinates of the points. Under simplified camera models, the problem reduces to factorizing a measurement matrix into the product of two low rank matrices. Each element of the measurement matrix contains the position of a point in a particular image. When all elements are observed, the problem can be solved trivially using SVD, but in any realistic situation many elements of the matrix are missing and the ones that are observed have a different directional uncertainty. Under these conditions, most existing factorization algorithms fail while human perception is relatively unchanged.

In this paper we use the well known EM algorithm for factor analysis to perform factorization. This allows us to easily handle missing data and measurement uncertainty and more importantly allows us to place a prior on the temporal trajectory of the latent variables (the camera position). We show that incorporating this prior gives a significant improvement in performance in challenging image sequences.

## 1 Introduction

Figure 1 illustrates the classical structure from motion (SFM) displays introduced by Ullman [13]. A transparent cylinder with painted dots rotates around its elongated axis. Even though no structure is apparent in any single frame, humans obtain a vivid percept of a cylinder[1].

SFM has been dealt with extensively in the computer vision literature. Typically a small number of feature points are tracked and a measurement matrix is formed in

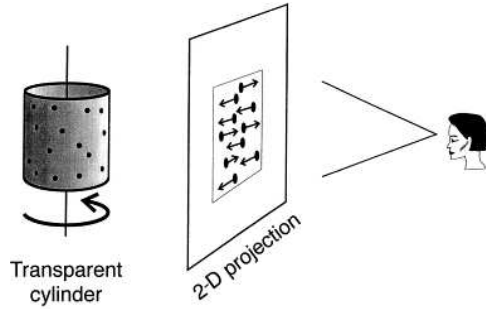

Transparent
cylinder

2-D projection

Figure 1: The classical structure from motion stimulus introduced by Ullman [13]. Humans continue to perceive the correct structure even when each dot appears only for a small number of frames, but most existing factorization algorithm fail in this case. Replotted from [1]

which each element corresponds to the image coordinates of a tracked point. The goal is to recover the camera motion and the $3D$ location of these points. Under simplified camera models it can be shown that this problem reduces to a problem of matrix factorization. We wish to describe the measurement matrix as a product of two low rank matrices. Thus if all features are reliably tracked in all the frames, the problem can be solved trivially using SVD [11]. In particular, performing an SVD on the measurement matrix of the rotating cylinder stimulus recovers the correct structure even if the measurement matrix is contaminated with significant amounts of noise and if the number of frames is relatively small.

But in any realistic situation, the measurement matrix will have *missing* entries. This is either because certain feature points are occluded in some of the frames and hence their positions are unknown, or due to a failure in the tracking algorithm. This has lead to the development of a number of algorithms for factorization with missing data [11, 6, 9, 2].

Factorization with missing data turns out to be much more difficult than the full data case. To illustrate the difficulty, consider the cylinder stimulus in figure 1. Humans still obtain a vivid percept of a cylinder even when each dot has a short "dot life". That is, each dot appears at a random starting frame, continues to appear for a small number of frames, and then disappears [12]. We applied the algorithms in [11, 6, 9, 2] to a sequence of 20 frames of a rotating cylinder in which the dot life was 10 frames. Thus the matrix was half full (or half empty). Surprisingly, none of the algorithms could recover the cylinder structure. They either failed to find any structure or they gave a structure that was drastically different from a cylinder. Presumably, humans are using additional prior knowledge that the algorithms are not.

In this paper we point out a source of information in image sequences that is usually neglected by factorization algorithms: temporal coherence. In a video sequence, the camera location at time $t + 1$ will probably be similar to its location at time $t$. In other words, if we randomly permute the temporal order of the frames, we will get a very unlikely image sequence. Yet nearly all existing factorization algorithms will be invariant to this random permutation of the frames: they only seek a low rank approximation to a matrix and permuting the rows of the matrix will not change the approximation.

In order to enable the use of temporal coherence, we formulate factorization in

terms of maximum likelihood for a factor analysis model, where the latent variable corresponds to camera position. We use the familiar EM algorithm for factor analysis to perform factorization with missing data and uncertainty. We show how to add a temporal coherence prior to the model and derive the EM updates. We show that incorporating this prior gives a significant improvement in performance in challenging image sequences.

## 2 Model

A set of $P$ feature points in $F$ images are tracked along an image sequence. Let $(u_{fp}, v_{fp})$ denote image coordinates of feature point $p$ in frame $f$. Let $U = (u_{fp})$, $V = (v_{fp})$ and $W = (w_{ij})$ where $w_{2i-1,j} = u_{ij}$ and $w_{2i,j} = v_{ij}$ for $1 \leq i \leq F$, i.e. $W$ is an interleaving of the rows of $U$ and $V$.

In the orthographic camera model, points in the $3D$ world are projected in parallel onto the image plane. For example, if the camera's optical center is in the origin (w.r.t $3D$ coordinate system), and its $x, y$ axes coincide with $X, Y$ axes in the $3D$ world, then taking a picture is a simple projection (in homogeneous coordinates):

$(x,y) = \begin{bmatrix} 1 & 0 & 0 & 0 \\ 0 & 1 & 0 & 0 \end{bmatrix} \begin{bmatrix} X \\ Y \\ Z \\ 1 \end{bmatrix}$. The depth, $Z$, has no influence on the image. In this

model, a camera can undergo rotation, translation, or a combination of the two.

Under orthography, and in the absence of noise,

$$[W]_{2F \times P} = [M]_{2F \times 4} [S]_{4 \times P} \tag{1}$$

where $M = \begin{bmatrix} M_1 \\ \vdots \\ M_F \end{bmatrix}_{2F \times 4}$ and $S = \begin{bmatrix} X_1 & \cdots & X_P \\ Y_1 & \cdots & Y_P \\ Z_1 & \cdots & Z_P \\ 1 & \cdots & 1 \end{bmatrix}_{4 \times P}$. $M$ describes camera

motion (rotation and translation, $[M_i]_{2 \times 4} = \begin{bmatrix} m_i^T & d_i \\ n_i^T & e_i \end{bmatrix}$ ). $m_i$ and $n_i$ are $3 \times 1$ vectors that describe the rotation of the camera; $d_i$ and $e_i$ are scalars describing camera translation, [2] and $S$ describes points location in $3D$.

For noisy observations, the model becomes:
$$[W]_{2F \times P} = [M]_{2F \times 4} [S]_{4 \times P} + [\eta]_{2F \times P} \tag{2}$$
where $\eta$ is Gaussian noise.

If the elements of the noise matrix $\eta$ are uncorrelated and of equal variance then we seek a factorization that minimizes the mean squared error between $W$ and $MS$. This can be solved trivially using the SVD of $W$. Missing data can be modeled using equation 2 by assuming some elements of the noise matrix $\eta$ have infinite variance. Obviously the SVD is not the solution once we allow different elements of $\eta$ to have different variances.

### 2.1 Factorization as factor analysis

It is well known that the SVD calculation can be formulated as a limiting case of maximum likelihood factor analysis [8]. In standard factor analysis we have a set

of observations $\{y(t)\}$ that are linear combinations of a latent variable $x(t)$:

$$y(t) = Ax(t) + \eta(t) \tag{3}$$

with $x(t) \sim N(0, \sigma_x^2 I)$ and $\eta(t) \sim N(0, \Psi_t)$. If $\Psi_t$ is a diagonal matrix with constant elements $\Psi_t = \sigma^2 I$ then in the limit $\sigma/\sigma_x \to 0$ the ML estimate for $A$ will give the same answer as the SVD. We now show how to rewrite the SFM problem in this form.

In equation 1 the horizontal and vertical coordinates of the same point appear in different rows. It can be rewritten as:

$$[U\ V]_{F \times 2P} = [M\ N]_{F \times 8} \begin{bmatrix} S & 0 \\ 0 & S \end{bmatrix}_{8 \times 2P} + [\tilde{\eta}]_{F \times 2P} \tag{4}$$

Let $y(t)$ be the vector of noisy observations (noisy image locations) at time $t$, i.e. $y(t) = [u(t)\,v(t)]$, that is $y(t) = [u_1(t), \cdots u_P(t)\, v_1(t), \cdots v_P(t)]^T$. Let $x(t)$ be a vector of length 8 that denotes the camera position at time $t$ $x(t) = [m(t)^T\, d(t)\, n(t)^T\, e(t)]^T$ and let $A = \begin{bmatrix} S^T & 0 \\ 0 & S^T \end{bmatrix}$. Identifying $y(t)$ with the $t$th row of the matrix $[U\ V]$ and $x(t)$ with the $t$th row of $[m\,n]$, then equation 4 is equivalent to equation 3.

We can now use the standard EM algorithm for factor analysis to find the ML estimate for $S$.

E step:

$$E(x(t)|y(t)) = \left(\sigma_x^{-2}I + A^T \Psi_t^{-1} A\right)^{-1} A^T \Psi_t^{-1} y(t) \tag{5}$$

$$V(x(t)|y(t)) = \left(\sigma_x^{-2}I + A^T \Psi_t^{-1} A\right)^{-1} \tag{6}$$

$$<x(t)> = E(x(t)|y(t)) \tag{7}$$

$$<x(t)x(t)^T> = V(x(t)|y(t)) - <x(t)><x(t)>^T \tag{8}$$

M step: In the M step we solve the normal equations for the structure $S$. The exact form depends on the structure of $\Psi_t$. Denote by $s_p$ a vector of length 3 that denotes the $3D$ coordinates of point $p$ then for a diagonal noise covariance matrix $\Psi_t$ the M step is:

$$s_p = B_p C_p^{-1} \tag{9}$$

where

$$B_p = \sum_t \left[ \Psi_t^{-1}(p,p)(u_{tp} - <d_t>) <m(t)^T > \right. \tag{10}$$
$$\left. + \Psi_t^{-1}(p+P, p+P)(v_{tp} - <e_t>) <n(t)>^T \right]$$
$$C_p = \sum_t \left[ \Psi_t^{-1}(p,p) <m(t)m(t)^T > \right.$$
$$\left. + \Psi_t^{-1}(p+P, p+P) <n(t)n(t)^T > \right]$$

where the expectation required in the $M$ step are the appropriate subvectors and submatrices of $<x(t)>$ and $<x(t)x(t)^T>$.

If we set $\Psi_t^{-1}(p,p) = \Psi_t^{-1}(p+P, p+P) = 0$ if point $p$ is missing in frame $t$ then we obtain an EM algorithm for factorization with missing data. Note that the form of the updates means we can put any value we wish in the missing elements of $y$ and they will be ignored by the algorithm.

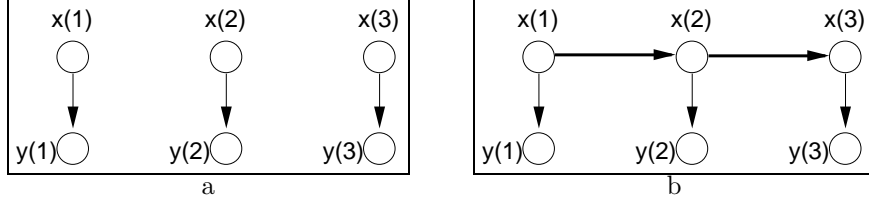

Figure 2: **a.** The graphical model assumed by most factorization algorithms for SFM. The camera location $x(t)$ is assumed to be independent of the camera location at any other time step. **b.** The graphical model assumed by our approach. We model temporal coherence by assuming a Markovian structure on the camera location.

A more realistic noise model for real images is that $\Psi_t$ is *not diagonal* but rather that the noise in the horizontal and vertical coordinates of the same point are correlated with an arbitrary $2 \times 2$ inverse covariance matrix. This problem is usually called *factorization with uncertainty* [5, 7]. It is easy to derive the M step in this case as well. It is similar to equation 9 except that cross terms involving $\Psi_t^{-1}(p, p + P)$ are also involved:

$$s_p \quad = \quad (B_p + B'_p)(C_p + C'_p)^{-1} \tag{11}$$

where

$$B'_p \quad = \quad \sum_t \left[ \Psi_t^{-1}(p, p + P)(v_{tp} - <e_t>) <m(t)^T> \right. \tag{12}$$

$$+ \ \Psi_t^{-1}(p + P, p)(u_{tp} - <d_t>) <n(t)>^T \big]$$

$$C'_p \quad = \quad \sum_t \left[ \Psi_t^{-1}(p, p + P) <n(t)m(t)^T> \right.$$

$$+ \ \Psi_t^{-1}(p + P, p) <m(t)n(t)^T> \big]$$

Regardless of uncertainty and missing data the complexity of the EM algorithm grows linearly with the number of feature points and the number of frames. At every iteration, the most computationally intensive step is an inversion of an $8 \times 8$ matrix.

## 2.2 Adding temporal coherence

The factor analysis algorithm for factorization assumes that the latent variables $x(t)$ are independent. In SFM this assumption means that the camera location in different frames is independent and hence permuting the order of the frames makes no difference for the factorization. As mentioned in the introduction, in almost any video sequence this assumption is wrong. Typically camera location varies smoothly as a function of time.

Figure 2a shows the graphical model corresponding to most factorization algorithms: the independence of the camera location is represented by the fact that every time step is isolated from the other time steps in the graph. But it is easy to fix this assumption by adding edges between the latent variables as shown in figure 2b.

Specifically, we use a second order approximation to the motion of the camera:

$$x(t) \quad = \quad x(t-1) + v(t-1) + \frac{1}{2}a(t-1) + \epsilon_1 \tag{13}$$

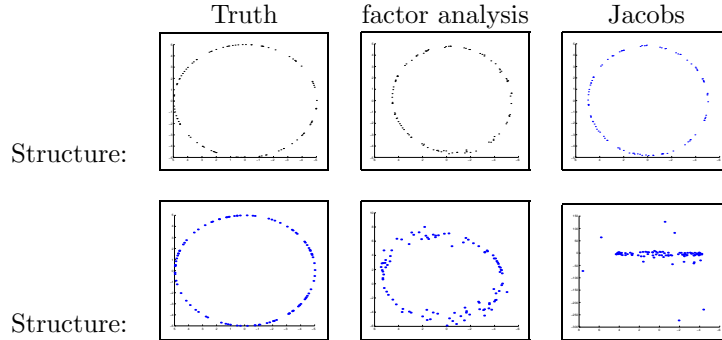

Figure 3: Comparison of factor analysis and Jacobs' algorithm on synthetic sequences. All other existing algorithms performed worse than Jacobs. They all fail when there is noise and missing data while factor analysis with temporal coherence succeeds. Structure and motion are shown from a top view.

$$
\begin{aligned}
v(t) &= v(t-1) + a(t-1) + \epsilon_2 & (14)\\
a(t) &= a(t-1) + \epsilon_3 & (15)\\
y(t) &= Ax(t) + \eta(t) & (16)
\end{aligned}
$$

Note that we do not assume that the $2D$ trajectory of each point is smooth. Rather we assume the $3D$ trajectory of the camera is smooth.

It is straightforward to derive the EM iterations for a ML estimate of $S$ using the model in equation 16. The M step is unchanged from the classical factor analysis and is given by equation 9. The only change in the E step is that $E(x(t)|y)$ and $V(x(t)|y)$ need to be calculated using a Kalman smoother. We use a standard RTS smoother [4]. Note that the computation of the E step is still linear in the number of frames and datapoints.

Kalman filtering has been used extensively in a more perspective SFM setting(e.g. [10]). However, in perspective projections the problem is no longer one of factorization. Thus even for Gaussian noise, the Extended Kalman filter needs to be used, smoothing is not performed and no guarantee of increase in likelihood is obtained. Within the factorization framework, we can use the classical Kalman filter and obtain a simple algorithm that provably increases the likelihood at every iteration.

## 3  Experiments

In this section we describe the experimental performance of EM with time coherence compared to ground truth and to previous algorithms for structure from motion with missing data [11, 6, 9, 2]. For [11, 6, 9] we used the Matlab implementation made public by D. Jacobs.

The first input sequence is the sequence of the cylinder shown in figure 1. 100 points uniformly drawn from the cylinder surface are tracked over 20 frames. Each of the points appears for 10 frames, starting at a random time, and then disappears. The observed image locations were added a Gaussian noise with standard deviation $\sigma = 0.1$.

We checked the performance of the different algorithms in the cases of: (1) full noise free observation matrix , (2) noisy full observation matrix, (3) noiseless observations

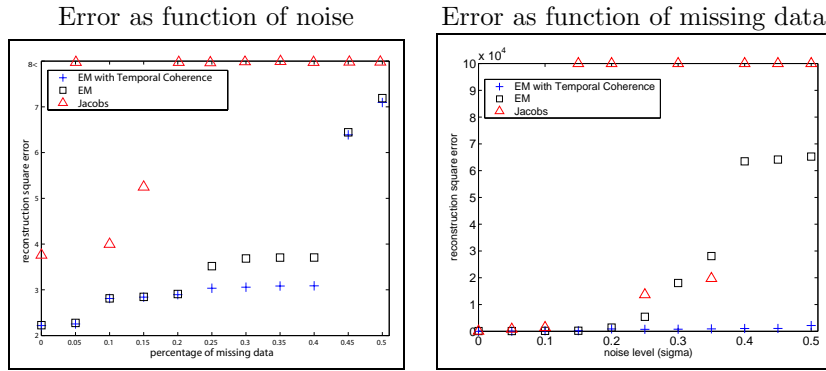

Figure 4: Graphs depict influence of noise and percentage of missing data on reconstruction results of factor analysis and [6].

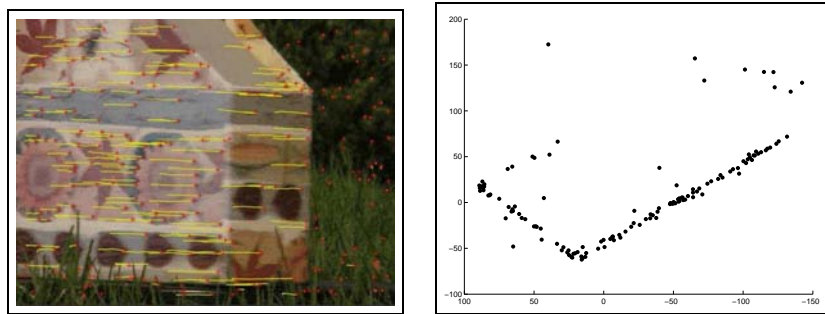

Figure 5: Results of scene reconstruction from a real sequence: A binder and is placed on a rotating surface filmed with a static camera. Our algorithm succeeded in (approximately) obtaining the right structure and all other algorithms failed. Results are shown in top view.

with missing data and (4) noisy observations with missing data.

All algorithms performed well and gave similar results for the full matrix noiseless sequence.

In the fully observed noisy case, factor analysis without temporal coherence gave comparable performance to Tomasi-Kanade, which minimize $\|MS - W\|_F^2$. When temporal coherence was added, the reconstruction results were improved. The results of Shum's algorithm were similar to Tomasi-Kanade. The algorithms of Jacobs and Brand turned to be noise sensitive.

In the case of noiseless missing data (figure 3 top), our algorithm and Jacobs' algorithm reconstruct the correct motion and structure. Tomasi-Kanade's algorithm and Shum's algorithm could not handle this pattern of missing data and failed to give any structure.

Once we add even very mild amounts of noise (figure 3 middle) all existing algorithms fail. While factor analysis with temporal coherence continues to extract the correct structure even for significant noise values.

Figure 5 shows result on a real sequence.

# 4   Discussion

Despite progress in algorithms for factorization with uncertainty the best existing algorithms still fall far short of human performance, even for seemingly simple stimuli. Presumably, humans are using additional prior information. In this paper we have focused on one particular prior: the temporal smoothness of the camera motion. We showed how to formulate SFM as a factor analysis problem and how to add temporal coherence to the EM algorithm. Our experimental results show that this simple prior can give a significant improvement in performance in challenging sequences.

Temporal coherence is just one of many possible priors. It has been suggested that humans also use a smoothness prior on the 3D surface they are perceiving [12]. It would be interesting to extend our framework in this direction.

The most drastic simplification our model makes is the assumption of Gaussian noise. It would be interesting to extend the algorithm to non Gaussian settings. This may require approximate inference algorithms in the E step as used in [3].

## Footnotes

[1]An online animation of this famous stimulus is available at: aris.ss.uci.edu/cogsci/personnel/hoffman/cylinderapplet.html

[2]We do not subtract the mean of each row from it, since in case of missing data the centroids of points do not coincide.

# References

[1] R.A. Andersen and D.C Bradley. Perception of three-dimensional structure from motion. In *Trends in Cognitive Sciences, 2*, pages 222–228, 1998.

[2] M.E. Brand. Incremental singular value decomposition of uncertain data with missing values. In *ECCV*, pages 707–720, May 2002.

[3] F. Dellaert, S. M. Seitz, C. E. Thorpe, and S. Thrun. Structure from motion without correspondence. In *ICCV*, pages 696–702, January 1999.

[4] Arthur Gelb, editor. *Applied Optimal Estimation*. MIT Press, 1974.

[5] M. Irani and P. Anandan. Factorization with uncertainty. In *ECCV*, pages 959–966, January 2000.

[6] D. Jacobs. Linear fitting with missing data: Applications to structure-from-motion and to characterizing intensity images. In *CVPR*, pages 206–212, 1997.

[7] D. D. Morris and T. Kanade. A unified factorization algorithm for points, line segments and planes with uncertain models. In *ICCV*, pages 696–702, January 1999.

[8] S. Roweis. Em algorithms for pca and spca. In *NIPS*, pages 431–437, 1997.

[9] H. Y. Shum, K. Ikeuchi, and R. Reddy. Principal component analysis with missing data and its application to polyhedral object modeling. pages 854–867, September 1995.

[10] S. Soatto and P. Perona. Reducing structure from motion: a general framework for dynamic vision. *IEEE Trans. on Pattern Analysis and Machine Intelligence*, pages 943–960, 1999.

[11] C. Tomasi and T. Kanade. Shape and motion from image streams under orthography: A factorization method. *Int. J. of Computer Vision*, 9(2):137–154, November 1992.

[12] S. Treue, M. Husain, and R. Andersen. Human perception of structure from motion. *Vision Research*, 31:59–75, 1991.

[13] S. Ullman. *The interpertation of visual motion*. MIT Press, 1979.
